# Structured Prediction via the Extragradient Method

**Ben Taskar**
Computer Science
UC Berkeley, Berkeley, CA 94720
taskar@cs.berkeley.edu

**Simon Lacoste-Julien**
Computer Science
UC Berkeley, Berkeley, CA 94720
slacoste@cs.berkeley.edu

**Michael I. Jordan**
Computer Science and Statistics
UC Berkeley, Berkeley, CA 94720
jordan@cs.berkeley.edu

## Abstract

We present a simple and scalable algorithm for large-margin estimation of structured models, including an important class of Markov networks and combinatorial models. We formulate the estimation problem as a convex-concave saddle-point problem and apply the extragradient method, yielding an algorithm with linear convergence using simple gradient and projection calculations. The projection step can be solved using combinatorial algorithms for min-cost quadratic flow. This makes the approach an efficient alternative to formulations based on reductions to a quadratic program (QP). We present experiments on two very different structured prediction tasks: 3D image segmentation and word alignment, illustrating the favorable scaling properties of our algorithm.

## 1  Introduction

The scope of discriminative learning methods has been expanding to encompass prediction tasks with increasingly complex structure. Much of this recent development builds upon graphical models to capture sequential, spatial, recursive or relational structure, but as we will discuss in this paper, the structured prediction problem is broader still. For graphical models, two major approaches to discriminative estimation have been explored: (1) maximum conditional likelihood [13] and (2) maximum margin [6, 1, 20]. For the broader class of models that we consider here, the conditional likelihood approach is intractable, but the large margin formulation yields tractable convex problems.

We interpret the term *structured output model* very broadly, as a compact scoring scheme over a (possibly very large) set of combinatorial structures and a method for finding the highest scoring structure. In graphical models, the scoring scheme is embodied in a probability distribution over possible assignments of the prediction variables as a function of input variables. In models based on combinatorial problems, the scoring scheme is usually a simple sum of weights associated with vertices, edges, or other components of a structure; these weights are often represented as parametric functions of a set of features. Given training instances labeled by desired structured outputs (e.g., matchings) and a set of

features that parameterize the scoring function, the learning problem is to find parameters such that the highest scoring outputs are as close as possible to the desired outputs.

Example of prediction tasks solved via combinatorial optimization problems include bipartite and non-bipartite matching in alignment of 2D shapes [5], word alignment in natural language translation [14] and disulfide connectivity prediction for proteins [3]. All of these problems can be formulated in terms of a tractable optimization problem. There are also interesting subfamilies of graphical models for which large-margin methods are tractable whereas likelihood-based methods are not; an example is the class of Markov random fields with restricted potentials used for object segmentation in vision [12, 2].

Tractability is not necessarily sufficient to obtain algorithms that work effectively in practice. In particular, although the problem of large margin estimation can be formulated as a quadratic program (QP) in several cases of interest [2, 19], and although this formulation exploits enough of the problem structure so as to achieve a polynomial representation in terms of the number of variables and constraints, off-the-shelf QP solvers scale poorly with problem and training sample size for these models. To solve large-scale machine learning problems, researchers often turn to simple gradient-based algorithms, in which each individual step is cheap in terms of computation and memory. Examples of this approach in the structured prediction setting include the Structured Sequential Minimal Optimization algorithm [20, 18] and the Structured Exponentiated Gradient algorithm [4]. These algorithms are first-order methods for solving QPs arising from low-treewidth Markov random fields and other decomposable models. They are able to scale to significantly larger problems than off-the-shelf QP solvers. However, they are limited in scope in that they rely on dynamic programming to compute essential quantities such as gradients. They do not extend to models in which dynamic programming is not applicable, for example, to problems such as matchings and min-cuts.

In this paper, we present an estimation methodology for structured prediction problems that does not require a general-purpose QP solver. We propose a saddle-point formulation which allows us to exploit simple gradient-based methods [11] with linear convergence guarantees. Moreover, we show that the key computational step in these methods—a certain projection operation—inherits the favorable computational complexity of the underlying optimization problem. This important result makes our approach viable computationally. In particular, for matchings and min-cuts, projection involves a min-cost quadratic flow computation, a problem for which efficient, highly-specialized algorithms are available. We illustrate the effectiveness of this approach on two very different large-scale structured prediction tasks: 3D image segmentation and word alignment in translation.

## 2  Structured models

We begin by discussing two special cases of the general framework that we subsequently present: (1) a class of Markov networks used for segmentation, and (2) a bipartite matching model for word alignment. Despite significant differences in the setup for these models, they share the property that in both cases the problem of finding the highest-scoring output can be formulated as a linear program (LP).

**Markov networks.** We consider a special class of Markov networks, common in vision applications, in which inference reduces to a tractable min-cut problem [7]. Focusing on binary variables, $\mathbf{y} = \{y_1, \ldots, y_N\}$, and pairwise potentials, we define a joint distribution over $\{0, 1\}^N$ via $P(\mathbf{y}) \propto \prod_{j \in \mathcal{V}} \phi_j(y_j) \prod_{jk \in \mathcal{E}} \phi_{jk}(y_j, y_k)$, where $(\mathcal{V}, \mathcal{E})$ is an undirected graph, and where $\{\phi_j(y_j); j \in \mathcal{V}\}$ are the node potentials and $\{\phi_{jk}(y_j, y_k), jk \in \mathcal{E}\}$ are the edge potentials.

In image segmentation (see Fig. 1(a)), the node potentials capture local evidence about the label of a pixel or laser scan point. Edges usually connect nearby pixels in an image, and serve to correlate their labels. Assuming that such correlations tend to be *positive*

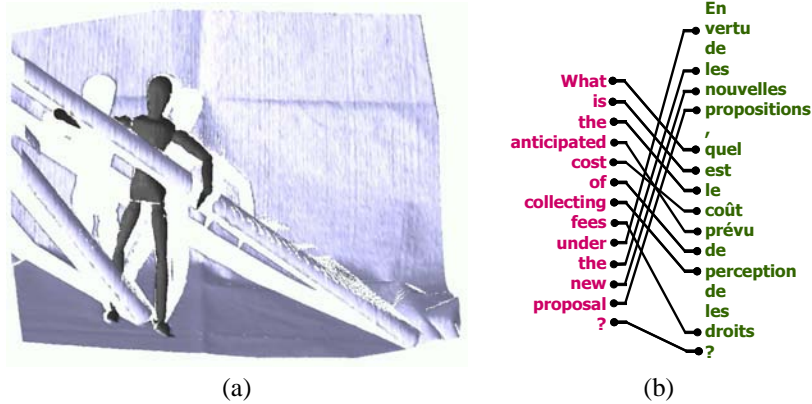

En
vertu
de
les
What
nouvelles
is
propositions
the
anticipated
cost
quel
of
est
collecting
le
fees
coût
under
prévu
the
de
new
perception
proposal
de
?
les
droits
?

(a)                                                    (b)

Figure 1: Examples of structured prediction applications: (a) articulated object segmentation and (b) word alignment in machine translation.

(connected nodes tend to have the same label), we restrict the form of edge potentials to be of the form $\phi_{jk}(y_j, y_k) = \exp\{-s_{jk}\mathbb{I}(y_j \neq y_k)\}$, where $s_{jk}$ is a non-negative penalty for assigning $y_j$ and $y_k$ different labels. Expressing node potentials as $\phi_j(y_j) = \exp\{s_j y_j\}$, we have $P(\mathbf{y}) \propto \exp\left\{\sum_{j \in \mathcal{V}} s_j y_j - \sum_{jk \in \mathcal{E}} s_{jk}\mathbb{I}(y_j \neq y_k)\right\}$. Under this restriction of the potentials, it is known that the problem of computing the maximizing assignment, $\mathbf{y}^* = \arg\max P(\mathbf{y} \mid \mathbf{x})$, has a tractable formulation as a min-cut problem [7]. In particular, we obtain the following LP:

$$\max_{0 \leq \mathbf{z} \leq 1} \sum_{j \in \mathcal{V}} s_j z_j - \sum_{jk \in \mathcal{E}} s_{jk} z_{jk} \quad \text{s.t.} \quad z_j - z_k \leq z_{jk}, \ z_k - z_j \leq z_{jk}, \ \forall jk \in \mathcal{E}. \quad (1)$$

In this LP, a continuous variable $z_j$ is a relaxation of the binary variable $y_j$. Note that the constraints are equivalent to $|z_j - z_k| \leq z_{jk}$. Because $s_{jk}$ is positive, $z_{jk} = |z_k - z_j|$ at the maximum, which is equivalent to $\mathbb{I}(z_j \neq z_k)$ if the $z_j, z_k$ variables are binary. An integral optimal solution always exists, as the constraint matrix is totally unimodular [17] (that is, the relaxation is not an approximation).

We can parametrize the node and edge weights $s_j$ and $s_{jk}$ in terms of user-provided features $\mathbf{x}_j$ and $\mathbf{x}_{jk}$ associated with the nodes and edges. In particular, in 3D range data, $\mathbf{x}_j$ might be spin image features or spatial occupancy histograms of a point $j$, while $\mathbf{x}_{jk}$ might include the distance between points $j$ and $k$, the dot-product of their normals, etc. The simplest model of dependence is a linear combination of features: $s_j = \mathbf{w}_n^\top \mathbf{f}_n(\mathbf{x}_j)$ and $s_{jk} = \mathbf{w}_e^\top \mathbf{f}_e(\mathbf{x}_{jk})$, where $\mathbf{w}_n$ and $\mathbf{w}_e$ are node and edge parameters, and $\mathbf{f}_n$ and $\mathbf{f}_e$ are node and edge feature mappings, of dimension $d_n$ and $d_e$, respectively. To ensure non-negativity of $s_{jk}$, we assume the edge features $\mathbf{f}_e$ to be non-negative and restrict $\mathbf{w}_e \geq 0$. This constraint is easily incorporated into the formulation we present below. We assume that the feature mappings $\mathbf{f}$ are provided by the user and our goal is to estimate parameters $\mathbf{w}$ from labeled data. We abbreviate the score assigned to a labeling $\mathbf{y}$ for an input $\mathbf{x}$ as $\mathbf{w}^\top \mathbf{f}(\mathbf{x}, \mathbf{y}) = \sum_j y_j \mathbf{w}_n^\top \mathbf{f}_n(\mathbf{x}_j) - \sum_{jk \in \mathcal{E}} y_{jk} \mathbf{w}_e^\top \mathbf{f}_e(\mathbf{x}_{jk})$, where $y_{jk} = \mathbb{I}(y_j \neq y_k)$.

**Matchings.** Consider modeling the task of word alignment of parallel bilingual sentences (see Fig. 1(b)) as a maximum weight bipartite matching problem, where the nodes $\mathcal{V} = \mathcal{V}^s \cup \mathcal{V}^t$ correspond to the words in the "source" sentence ($\mathcal{V}^s$) and the "target" sentence ($\mathcal{V}^t$) and the edges $\mathcal{E} = \{jk : j \in \mathcal{V}^s, k \in \mathcal{V}^t\}$ correspond to possible alignments between them. For simplicity, assume that each word aligns to one or zero words in the other sentence. The edge weight $s_{jk}$ represents the degree to which word $j$ in one sentence can translate into the word $k$ in the other sentence. Our objective is to find an alignment that maximizes the sum of edge scores. We represent a matching using a set of binary variables

$y_{jk}$ that are set to 1 if word $j$ is assigned to word $k$ in the other sentence, and 0 otherwise. The score of an assignment is the sum of edge scores: $s(\mathbf{y}) = \sum_{jk \in \mathcal{E}} s_{jk} y_{jk}$. The maximum weight bipartite matching problem, $\arg\max_{\mathbf{y} \in \mathcal{Y}} s(\mathbf{y})$, can be found by solving the following LP:

$$\max_{0 \leq \mathbf{z} \leq 1} \sum_{jk \in \mathcal{E}} s_{jk} z_{jk} \quad \text{s.t.} \quad \sum_{j \in \mathcal{V}^s} z_{jk} \leq 1, \forall k \in \mathcal{V}^t; \quad \sum_{k \in \mathcal{V}^t} z_{jk} \leq 1, \forall j \in \mathcal{V}^s, \quad (2)$$

where again the continuous variables $z_{jk}$ correspond to the relaxation of the binary variables $y_{jk}$. As in the min-cut problem, this LP is guaranteed to have integral solutions for any scoring function $s(\mathbf{y})$ [17].

For word alignment, the scores $s_{jk}$ can be defined in terms of the word pair $jk$ and input features associated with $\mathbf{x}_{jk}$. We can include the identity of the two words, relative position in the respective sentences, part-of-speech tags, string similarity (for detecting cognates), etc. We let $s_{jk} = \mathbf{w}^\top \mathbf{f}(\mathbf{x}_{jk})$ for some user-provided feature mapping $\mathbf{f}$ and abbreviate $\mathbf{w}^\top \mathbf{f}(\mathbf{x}, \mathbf{y}) = \sum_{jk} y_{jk} \mathbf{w}^\top \mathbf{f}(\mathbf{x}_{jk})$.

**General structure.** More generally, we consider prediction problems in which the input $\mathbf{x} \in \mathcal{X}$ is an arbitrary structured object and the output is a vector of values $\mathbf{y} = (y_1, \ldots, y_{L_\mathbf{x}})$, for example, a matching or a cut in the graph. We assume that the length $L_\mathbf{x}$ and the structure of $\mathbf{y}$ depend deterministically on the input $\mathbf{x}$. In our word alignment example, the output space is defined by the length of the two sentences. Denote the output space for a given input $\mathbf{x}$ as $\mathcal{Y}(\mathbf{x})$ and the entire output space as $\mathcal{Y} = \bigcup_{\mathbf{x} \in \mathcal{X}} \mathcal{Y}(\mathbf{x})$.

Consider the class of structured prediction models $\mathcal{H}$ defined by the linear family: $h_\mathbf{w}(\mathbf{x}) = \arg\max_{\mathbf{y} \in \mathcal{Y}(\mathbf{x})} \mathbf{w}^\top \mathbf{f}(\mathbf{x}, \mathbf{y})$, where $\mathbf{f}(\mathbf{x}, \mathbf{y})$ is a vector of functions $\mathbf{f} : \mathcal{X} \times \mathcal{Y} \mapsto \mathbb{R}^n$. This formulation is very general. Indeed, it is too general for our purposes—for many $\mathbf{f}, \mathcal{Y}$ pairs, finding the optimal $\mathbf{y}$ is intractable. Below, we specialize to the class of models in which the $\arg\max$ problem can be solved in polynomial time using linear programming (and more generally, convex optimization); this is still a very large class of models.

## 3 Max-margin estimation

We assume a set of training instances $S = \{(\mathbf{x}_i, \mathbf{y}_i)\}_{i=1}^m$, where each instance consists of a structured object $\mathbf{x}_i$ (such as a graph) and a target solution $\mathbf{y}_i$ (such as a matching). Consider learning the parameters $\mathbf{w}$ in the conditional likelihood setting. We can define $P_\mathbf{w}(\mathbf{y} \mid \mathbf{x}) = \frac{1}{Z_\mathbf{w}(\mathbf{x})} \exp\{\mathbf{w}^\top \mathbf{f}(\mathbf{x}, \mathbf{y})\}$, where $Z_\mathbf{w}(\mathbf{x}) = \sum_{\mathbf{y}' \in \mathcal{Y}(\mathbf{x})} \exp\{\mathbf{w}^\top \mathbf{f}(\mathbf{x}, \mathbf{y}')\}$, and maximize the conditional log-likelihood $\sum_i \log P_\mathbf{w}(\mathbf{y}_i \mid \mathbf{x}_i)$, perhaps with additional regularization of the parameters $\mathbf{w}$. However, computing the partition function $Z_\mathbf{w}(\mathbf{x})$ is #P-complete [23, 10] for the two structured prediction problems we presented above, matchings and min-cuts. Instead, we adopt the max-margin formulation of [20], which directly seeks to find parameters $\mathbf{w}$ such that: $\mathbf{y}_i = \arg\max_{\mathbf{y}_i' \in \mathcal{Y}_i} \mathbf{w}^\top \mathbf{f}(\mathbf{x}_i, \mathbf{y}_i'), \quad \forall i$, where $\mathcal{Y}_i = \mathcal{Y}(\mathbf{x}_i)$ and $\mathbf{y}_i$ denotes the appropriate vector of variables for example $i$. The solution space $\mathcal{Y}_i$ depends on the structured object $\mathbf{x}_i$; for example, the space of possible matchings depends on the precise set of nodes and edges in the graph.

As in univariate prediction, we measure the error of prediction using a loss function $\ell(\mathbf{y}_i, \mathbf{y}_i')$. To obtain a convex formulation, we upper bound the loss $\ell(\mathbf{y}_i, h_\mathbf{w}(\mathbf{x}_i))$ using the hinge function: $\max_{\mathbf{y}_i' \in \mathcal{Y}_i}[\mathbf{w}^\top \mathbf{f}_i(\mathbf{y}_i') + \ell_i(\mathbf{y}_i')] - \mathbf{w}^\top \mathbf{f}_i(\mathbf{y}_i)$, where $\ell_i(\mathbf{y}_i') = \ell(\mathbf{y}_i, \mathbf{y}_i')$, and $\mathbf{f}_i(\mathbf{y}_i') = \mathbf{f}(\mathbf{x}_i, \mathbf{y}_i')$. Minimizing this upper bound will force the true structure $\mathbf{y}_i$ to be optimal with respect to $\mathbf{w}$ for each instance $i$. We add a standard $L_2$ weight penalty $\frac{||\mathbf{w}||^2}{2C}$:

$$\min_{\mathbf{w} \in \mathcal{W}} \frac{||\mathbf{w}||^2}{2C} + \sum_i \max_{\mathbf{y}_i' \in \mathcal{Y}_i}[\mathbf{w}^\top \mathbf{f}_i(\mathbf{y}_i') + \ell_i(\mathbf{y}_i')] - \mathbf{w}^\top \mathbf{f}_i(\mathbf{y}_i), \quad (3)$$

where $C$ is a regularization parameter and $\mathcal{W}$ is the space of allowed weights (for example, $\mathcal{W} = \mathrm{I\!R}^n$ or $\mathcal{W} = \mathrm{I\!R}_+^n$). Note that this formulation is equivalent to the standard formulation using slack variables $\xi$ and slack penalty $C$ presented in [20, 19].

The key to solving Eq. (3) efficiently is the *loss-augmented inference problem*, $\max_{\mathbf{y}_i' \in \mathcal{Y}_i}[\mathbf{w}^\top \mathbf{f}_i(\mathbf{y}_i') + \ell_i(\mathbf{y}_i')]$. This optimization problem has precisely the same form as the prediction problem whose parameters we are trying to learn—$\max_{\mathbf{y}_i' \in \mathcal{Y}_i} \mathbf{w}^\top \mathbf{f}_i(\mathbf{y}_i')$— but with an additional term corresponding to the loss function. Tractability of the loss-augmented inference thus depends not only on the tractability of $\max_{\mathbf{y}_i' \in \mathcal{Y}_i} \mathbf{w}^\top \mathbf{f}_i(\mathbf{y}_i')$, but also on the form of the loss term $\ell_i(\mathbf{y}_i')$. A natural choice in this regard is the Hamming distance, which simply counts the number of variables in which a candidate solution $\mathbf{y}_i'$ differs from the target output $\mathbf{y}_i$. In general, we need only assume that the loss function decomposes over the variables in $\mathbf{y}_i$.

For example, in the case of bipartite matchings the Hamming loss counts the number of different edges in the matchings $\mathbf{y}_i$ and $\mathbf{y}_i'$ and can be written as: $\ell_i^H(\mathbf{y}_i') = \sum_{jk} y_{i,jk} + \sum_{jk}(1 - 2y_{i,jk}')y_{i,jk}$. Thus the loss-augmented matching problem for example $i$ can be written as an LP similar to Eq. (2) (without the constant term $\sum_{jk} y_{i,jk}$):

$$\max_{0 \leq \mathbf{z} \leq 1} \quad \sum_{jk} z_{i,jk}[\mathbf{w}^\top \mathbf{f}(\mathbf{x}_{i,jk}) + 1 - 2y_{i,jk}] \quad \text{s.t.} \quad \sum_j z_{i,jk} \leq 1, \quad \sum_k z_{i,jk} \leq 1.$$

Generally, when we can express $\max_{\mathbf{y}_i' \in \mathcal{Y}_i} \mathbf{w}^\top \mathbf{f}_i(\mathbf{y}_i')$ as an LP, $\max_{\mathbf{z}_i \in \mathcal{Z}_i} \mathbf{w}^\top \mathbf{F}_i \mathbf{z}_i$, where $\mathcal{Z}_i = \{\mathbf{z}_i : \mathbf{A}_i \mathbf{z}_i \leq \mathbf{b}_i, \ \mathbf{z}_i \geq 0\}$, for appropriately defined constraints $\mathbf{A}_i, \mathbf{b}_i$ and feature matrix $\mathbf{F}_i$, we have a similar LP for the loss-augmented inference for each example $i$: $d_i + \max_{\mathbf{z}_i \in \mathcal{Z}_i}(\mathbf{w}^\top \mathbf{F}_i + \mathbf{c}_i)^\top \mathbf{z}_i$ for appropriately defined $d_i, \mathbf{F}_i, \mathbf{c}_i, \mathbf{A}_i, \mathbf{b}_i$. Let $\mathbf{z} = \{\mathbf{z}_1, \ldots, \mathbf{z}_m\}$, $\mathcal{Z} = \mathcal{Z}_1 \times \ldots \times \mathcal{Z}_m$.

We could proceed by making use of Lagrangian duality, which yields a joint convex optimization problem; this is the approach described in [19]. Instead we take a different tack here, posing the problem in its natural saddle-point form:

$$\min_{\mathbf{w} \in \mathcal{W}} \max_{\mathbf{z} \in \mathcal{Z}} \quad \frac{||\mathbf{w}||^2}{2C} + \sum_i \left[ \mathbf{w}^\top \mathbf{F}_i \mathbf{z}_i + \mathbf{c}_i^\top \mathbf{z}_i - \mathbf{w}^\top \mathbf{f}_i(\mathbf{y}_i) \right]. \tag{4}$$

As we discuss in the following section, this approach allows us to exploit the structure of $\mathcal{W}$ and $\mathcal{Z}$ *separately*, allowing for efficient solutions for a wider range of structure spaces.

## 4 Extragradient method

The key operations of the method we present below are gradient calculations and Euclidean projections. We let $L(\mathbf{w}, \mathbf{z}) = \frac{||\mathbf{w}||^2}{2C} + \sum_i \left[ \mathbf{w}^\top \mathbf{F}_i \mathbf{z}_i + \mathbf{c}_i^\top \mathbf{z}_i - \mathbf{w}^\top \mathbf{f}_i(\mathbf{y}_i) \right]$, with gradients given by: $\nabla_\mathbf{w} L(\mathbf{w}, \mathbf{z}) = \frac{\mathbf{w}}{C} + \sum_i \mathbf{F}_i \mathbf{z}_i - \mathbf{f}_i(\mathbf{y}_i)$ and $\nabla_{\mathbf{z}_i} L(\mathbf{w}, \mathbf{z}) = \mathbf{F}_i^\top \mathbf{w} + \mathbf{c}_i$. We denote the projection of a vector $\mathbf{z}_i$ onto $\mathcal{Z}_i$ as $\pi_{\mathcal{Z}_i}(\mathbf{z}_i) = \arg\min_{\mathbf{z}_i' \in \mathcal{Z}_i} ||\mathbf{z}_i' - \mathbf{z}_i||$ and similarly, the projection onto $\mathcal{W}$ as $\pi_{\mathcal{W}}(\mathbf{w}') = \arg\min_{\mathbf{w} \in \mathcal{W}} ||\mathbf{w}' - \mathbf{w}||$.

A well-known solution strategy for saddle-point optimization is provided by the *extragradient method* [11]. An iteration of the extragradient method consists of two very simple steps, prediction $(\mathbf{w}, \mathbf{z}) \rightarrow (\mathbf{w}^p, \mathbf{z}^p)$ and correction $(\mathbf{w}^p, \mathbf{z}^p) \rightarrow (\mathbf{w}^c, \mathbf{z}^c)$:

$$\mathbf{w}^p = \pi_{\mathcal{W}}(\mathbf{w} - \beta \nabla_\mathbf{w} L(\mathbf{w}, \mathbf{z})); \qquad \mathbf{z}_i^p = \pi_{\mathcal{Z}_i}(\mathbf{z}_i + \beta \nabla_{\mathbf{z}_i} L(\mathbf{w}, \mathbf{z})); \tag{5}$$

$$\mathbf{w}^c = \pi_{\mathcal{W}}(\mathbf{w} - \beta \nabla_\mathbf{w} L(\mathbf{w}^p, \mathbf{z}^p)); \qquad \mathbf{z}_i^c = \pi_{\mathcal{Z}_i}(\mathbf{z}_i + \beta \nabla_{\mathbf{z}_i} L(\mathbf{w}^p, \mathbf{z}^p)); \tag{6}$$

where $\beta$ is an appropriately chosen step size. The algorithm starts with a feasible point $\mathbf{w} = 0$, $\mathbf{z}_i$'s that correspond to the assignments $\mathbf{y}_i$'s and step size $\beta = 1$. After each prediction step, it computes $r = \beta \frac{||\nabla L(\mathbf{w}, \mathbf{z}) - \nabla L(\mathbf{w}^p, \mathbf{z}^p)||}{(||\mathbf{w} - \mathbf{w}^p|| + ||\mathbf{z} - \mathbf{z}^p||)}$. If $r$ is greater than a threshold $\nu$, the

step size is decreased using an Armijo type rule: $\beta = (2/3)\beta \min(1, 1/r)$, and a new prediction step is computed until $r \leq \nu$, where $\nu \in (0, 1)$ is a parameter of the algorithm. Once a suitable $\beta$ is found, the correction step is taken and $(\mathbf{w}^c, \mathbf{z}^c)$ becomes the new $(\mathbf{w}, \mathbf{z})$. The method is guaranteed to converge linearly to a solution $\mathbf{w}^*, \mathbf{z}^*$ [11, 9]. See the longer version of this paper at `http://www.cs.berkeley.edu/~taskar/extragradient.pdf` for details. By comparison, Exponentiated Gradient [4] has sublinear convergence rate guarantees, while Structured SMO [18] has none.

The key step influencing the efficiency of the algorithm is the Euclidean projection onto the feasible sets $\mathcal{W}$ and $\mathcal{Z}_i$. In case $\mathcal{W} = \mathbb{R}^n$, the projection is the identity operation; projecting onto $\mathbb{R}^n_+$ consists of clipping negative weights to zero. Additional problem-specific constraints on the weight space can be efficiently incorporated in this step (although linear convergence guarantees only hold for polyhedral $\mathcal{W}$). In case of word alignment, $\mathcal{Z}_i$ is the convex hull of bipartite matchings and the problem reduces to the much-studied minimum cost quadratic flow problem. The projection $\mathbf{z}_i = \pi_{\mathcal{Z}_i}(\mathbf{z}'_i)$ is given by

$$\min_{0 \leq \mathbf{z} \leq 1} \quad \sum_{jk} \frac{1}{2}(z'_{i,jk} - z_{i,jk})^2 \quad \text{s.t.} \quad \sum_j z_{i,jk} \leq 1, \quad \sum_k z_{i,jk} \leq 1.$$

We use a standard reduction of bipartite matching to min-cost flow by introducing a source node $s$ linked to all the nodes in $\mathcal{V}_i^s$ (words in the "source" sentence), and a sink node $t$ linked from all the nodes in $\mathcal{V}_i^t$ (words in the "target" sentence), using edges of capacity 1 and cost 0. The original edges $jk$ have a quadratic cost $\frac{1}{2}(z'_{i,jk} - z_{i,jk})^2$ and capacity 1. Minimum (quadratic) cost flow from $s$ to $t$ is the projection of $\mathbf{z}'_i$ onto $\mathcal{Z}_i$.

The reduction of the projection to minimum quadratic cost flow for the min-cut polytope $\mathcal{Z}_i$ is shown in the longer version of the paper. Algorithms for solving this problem are nearly as efficient as those for solving regular min-cost flow problems. In case of word alignment, the running time scales with the cube of the sentence length. We use publicly-available code for solving this problem [8] (see `http://www.math.washington.edu/~tseng/netflowg_nl/`).

## 5 Experiments

We investigate two structured models we described above: bipartite matchings for word alignments and restricted potential Markov nets for 3D segmentation. A commercial QP-solver, MOSEK, runs out of memory on the problems we describe below using the QP formulation [19]. We compared the extragradient method with the averaged perceptron algorithm [6]. A question which arises in practice is how to choose the regularization parameter $C$. The typical approach is to run the algorithm for several values of the regularization parameter and pick the best model using a validation set. For the averaged perceptron, a standard method is to run the algorithm tracking its performance on a validation set, and selecting the model with best performance. We use the same training regime for the extragradient by running it with $C = \infty$.

**Object segmentation.** We test our algorithm on a 3D scan segmentation problem using the class of Markov networks with potentials that were described above. The dataset is a challenging collection of cluttered scenes containing articulated wooden puppets [2]. It contains eleven different single-view scans of three puppets of varying sizes and positions, with clutter and occluding objects such as rope, sticks and rings. Each scan consists of around $7,000$ points. Our goal was to segment the scenes into two classes— puppet and background. We use five of the scenes for our training data, three for validation and three for testing. Sample scans from the training and test set can be seen at `http://www.cs.berkeley.edu/~taskar/3DSegment/`. We computed spin images of size $10 \times 5$ bins at two different resolutions, then scaled the values and performed PCA to obtain 45 principal components, which comprised our node features. We used the surface links output by the scanner as edges between points and for each edge only used a

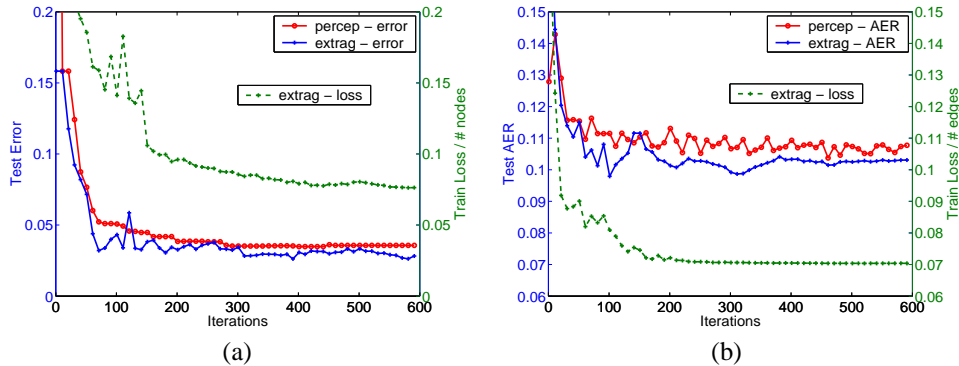

Figure 2: Both plots show test error for the averaged perceptron and the extragradient (left y-axis) and training loss per node or edge for the extragradient (right y-axis) versus number of iterations for (a) object segmentation task and (b) word alignment task.

single feature, set to a constant value of 1 for all edges. This results in all edges having the same potential. The training data contains approximately $37,000$ nodes and $88,000$ edges. Training time took about 4 hours for 600 iterations on a 2.80GHz Pentium 4 machine. Fig. 2(a) shows that the extragradient has a consistently lower error rate (about 3% for extragradient, 4% for averaged perceptron), using only slightly more expensive computations per iteration. Also shown is the corresponding decrease in the hinge-loss upperbound on the training data as the extragradient progresses.

**Word alignment.** We also tested our learning algorithm on word-level alignment using a data set from the 2003 NAACL set [15], the English-French Hansards task. This corpus consists of 1.1M automatically aligned sentences, and comes with a validation set of 39 sentence pairs and a test set of 447 sentences. The validation and test sentences have been hand-aligned and are marked with both *sure* and *possible* alignments. Using these alignments, *alignment error rate* (AER) is calculated as: $AER(A, S, P) = 1 - \frac{|A \cap S| + |A \cap P|}{|A| + |S|}$. Here, $A$ is a set of proposed index pairs, $S$ is the set of sure gold pairs, and $P$ is the set of possible gold pairs (where $S \subseteq P$).

We used the intersection of the predictions of the English-to-French and French-to-English IBM Model 4 alignments (using GIZA++ [16]) on the first 5000 sentence pairs from the 1.1M sentences. The number of edges for 5000 sentences was about 555,000. We tested on the 347 hand-aligned test examples, and used the validation set to select the stopping point. The features on the word pair $(e_j, f_k)$ include measures of association, orthography, relative position, predictions of generative models (see [22] for details). It took about 3 hours to perform 600 training iterations on the training data using a 2.8GHz Pentium 4 machine. Fig. 2(b) shows the extragradient performing slightly better (by about 0.5%) than average perceptron.

## 6 Conclusion

We have presented a general solution strategy for large-scale structured prediction problems. We have shown that these problems can be formulated as saddle-point optimization problems, problems that are amenable to solution by the extragradient algorithm. Key to our approach is the recognition that the projection step in the extragradient algorithm can be solved by network flow algorithms. Network flow algorithms are among the most well-developed in the field of combinatorial optimization, and yield stable, efficient algorithmic platforms. We have exhibited the favorable scaling of this overall approach in two concrete, large-scale learning problems. It is also important to note that the general approach extends to a much broader class of problems. In [21], we show how to apply this approach efficiently to other types of models, including general Markov networks and weighted context-free grammars, using Bregman projections.

## Acknowledgments

We thank Paul Tseng for kindly answering our questions about his min-cost flow code. This work was funded by the DARPA CALO project (03-000219) and Microsoft Research MICRO award (05-081). SLJ was also supported by an NSERC graduate sholarship.

## References

[1] Y. Altun, I. Tsochantaridis, and T. Hofmann. Hidden Markov support vector machines. In *Proc. ICML*, 2003.

[2] D. Anguelov, B. Taskar, V. Chatalbashev, D. Koller, D. Gupta, G. Heitz, and A. Ng. Discriminative learning of Markov random fields for segmentation of 3d scan data. In *CVPR*, 2005.

[3] P. Baldi, J. Cheng, and A. Vullo. Large-scale prediction of disulphide bond connectivity. In *Proc. NIPS*, 2004.

[4] P. Bartlett, M. Collins, B. Taskar, and D. McAllester. Exponentiated gradient algorithms for large-margin structured classification. In *NIPS*, 2004.

[5] S. Belongie, J. Malik, and J. Puzicha. Shape matching and object recognition using shape contexts. *IEEE Trans. Pattern Anal. Mach. Intell.*, 24, 2002.

[6] M. Collins. Discriminative training methods for hidden Markov models: Theory and experiments with perceptron algorithms. In *Proc. EMNLP*, 2002.

[7] D. M. Greig, B. T. Porteous, and A. H. Seheult. Exact maximum a posteriori estimation for binary images. *J. R. Statist. Soc. B*, 51, 1989.

[8] F. Guerriero and P. Tseng. Implementation and test of auction methods for solving generalized network flow problems with separable convex cost. *Journal of Optimization Theory and Applications*, 115(1):113–144, October 2002.

[9] B.S. He and L. Z. Liao. Improvements of some projection methods for monotone nonlinear variational inequalities. *JOTA*, 112:111:128, 2002.

[10] M. Jerrum and A. Sinclair. Polynomial-time approximation algorithms for the Ising model. *SIAM J. Comput.*, 22, 1993.

[11] G. M. Korpelevich. The extragradient method for finding saddle points and other problems. *Ekonomika i Matematicheskie Metody*, 12:747:756, 1976.

[12] S. Kumar and M. Hebert. Discriminative fields for modeling spatial dependencies in natural images. In *NIPS*, 2003.

[13] J. Lafferty, A. McCallum, and F. Pereira. Conditional random fields: Probabilistic models for segmenting and labeling sequence data. In *ICML*, 2001.

[14] E. Matusov, R. Zens, and H. Ney. Symmetric word alignments for statistical machine translation. In *Proc. COLING*, 2004.

[15] R. Mihalcea and T. Pedersen. An evaluation exercise for word alignment. In *Proceedings of the HLT-NAACL 2003 Workshop, Building and Using parallel Texts: Data Driven Machine Translation and Beyond*, pages 1–6, Edmonton, Alberta, Canada, 2003.

[16] F. Och and H. Ney. A systematic comparison of various statistical alignment models. *Computational Linguistics*, 29(1), 2003.

[17] A. Schrijver. *Combinatorial Optimization: Polyhedra and Efficiency*. Springer, 2003.

[18] B. Taskar. *Learning Structured Prediction Models: A Large Margin Approach*. PhD thesis, Stanford University, 2004.

[19] B. Taskar, V. Chatalbashev, D. Koller, and C. Guestrin. Learning structured prediction models: a large margin approach. In *ICML*, 2005.

[20] B. Taskar, C. Guestrin, and D. Koller. Max margin Markov networks. In *NIPS*, 2003.

[21] B. Taskar, S. Lacoste-Julien, and M. Jordan. Structured prediction, dual extragradient and Bregman projections. Technical report, UC Berkeley Statistics Department, 2005.

[22] B. Taskar, S. Lacoste-Julien, and D. Klein. A discriminative matching approach to word alignment. In *EMNLP*, 2005.

[23] L. G. Valiant. The complexity of computing the permanent. *Theoretical Computer Science*, 8:189–201, 1979.
